# Parallel Support Vector Machines:
# The Cascade SVM

**Hans Peter Graf, Eric Cosatto,**
**Leon Bottou, Igor Durdanovic, Vladimir Vapnik**

NEC Laboratories
4 Independence Way, Princeton, NJ 08540
*{hpg, cosatto, leonb, igord, vlad}@nec-labs.com*

## Abstract

We describe an algorithm for support vector machines (SVM) that
can be parallelized efficiently and scales to very large problems with
hundreds of thousands of training vectors. Instead of analyzing the
whole training set in one optimization step, the data are split into
subsets and optimized separately with multiple SVMs. The partial
results are combined and filtered again in a 'Cascade' of SVMs, until
the global optimum is reached. The Cascade SVM can be spread over
multiple processors with minimal communication overhead and
requires far less memory, since the kernel matrices are much smaller
than for a regular SVM. Convergence to the global optimum is
guaranteed with multiple passes through the Cascade, but already a
single pass provides good generalization. A single pass is 5x – 10x
faster than a regular SVM for problems of 100,000 vectors when
implemented on a single processor. Parallel implementations on a
cluster of 16 processors were tested with over 1 million vectors
(2-class problems), converging in a day or two, while a regular SVM
never converged in over a week.

## 1  Introduction

Support Vector Machines [1] are powerful classification and regression tools, but
their compute and storage requirements increase rapidly with the number of training
vectors, putting many problems of practical interest out of their reach. The core of an
SVM is a quadratic programming problem (QP), separating support vectors from the
rest of the training data. General-purpose QP solvers tend to scale with the cube of the
number of training vectors ($O(k^3)$). Specialized algorithms, typically based on
gradient descent methods, achieve impressive gains in efficiency, but still become
impractically slow for problem sizes on the order of 100,000 training vectors (2-class
problems).

One approach for accelerating the QP is based on 'chunking' [2][3][4], where subsets
of the training data are optimized iteratively, until the global optimum is reached.
'Sequential Minimal Optimization' (SMO) [5], which reduces the chunk size to 2
vectors, is the most popular of these algorithms. Eliminating non-support vectors

early during the optimization process is another strategy that provides substantial savings in computation. Efficient SVM implementations incorporate steps known as 'shrinking' for identifying non-support vectors early [4][6][7]. In combination with caching of the kernel data, such techniques reduce the computation requirements by orders of magnitude. Another approach, named 'digesting' optimizes subsets closer to completion before adding new data [8], saving considerable amounts of storage.

Improving compute-speed through parallelization is difficult due to dependencies between the computation steps. Parallelizations have been proposed by splitting the problem into smaller subsets and training a network to assign samples to different subsets [9]. Variations of the standard SVM algorithm, such as the Proximal SVM have been developed that are better suited for parallelization [10], but how widely they are applicable, in particular to high-dimensional problems, remains to be seen. A parallelization scheme was proposed where the kernel matrix is approximated by a block-diagonal [11]. A technique called *variable projection method* [12] looks promising for improving the parallelization of the optimization loop.

In order to break through the limits of today's SVM implementations we developed a distributed architecture, where smaller optimizations are solved independently and can be spread over multiple processors, yet the ensemble is guaranteed to converge to the globally optimal solution.

## 2   The Cascade SVM

As mentioned above, eliminating non-support vectors early from the optimization proved to be an effective strategy for accelerating SVMs. Using this concept we developed a filtering process that can be parallelized efficiently. After evaluating multiple techniques, such as projections onto subspaces (in feature space) or clustering techniques, we opted to use SVMs as filters. This makes it straightforward to drive partial solutions towards the global optimum, while alternative techniques may optimize criteria that are not directly relevant for finding the global solution.

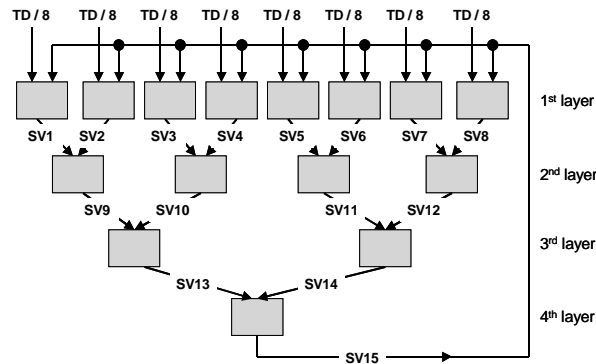

**Figure 1:** Schematic of a binary Cascade architecture. The data are split into subsets and each one is evaluated individually for support vectors in the first layer. The results are combined two-by-two and entered as training sets for the next layer. The resulting support vectors are tested for global convergence by feeding the result of the last layer into the first layer, together with the non-support vectors. TD: Training data, SVi: Support vectors produced by optimization i.

We initialize the problem with a number of independent, smaller optimizations and combine the partial results in later stages in a hierarchical fashion, as shown in Figure 1. Splitting the data and combining the results can be done in many different ways.

Figure 1 merely represents one possible architecture, a binary Cascade that proved to be efficient in many tests. It is guaranteed to advance the optimization function in every layer, requires only modest communication from one layer to the next, and converges to a good solution quickly.

In the architecture of Figure 1 sets of support vectors from two SVMs are combined and the optimization proceeds by finding the support vectors in each of the combined subsets. This continues until only one set of vectors is left. Often a single pass through this Cascade produces satisfactory accuracy, but if the global optimum has to be reached, the result of the last layer is fed back into the first layer. Each of the SVMs in the first layer receives all the support vectors of the last layer as inputs and tests its fraction of the input vectors, if any of them have to be incorporated into the optimization. If this is not the case for all SVMs of the input layer, the Cascade has converged to the global optimum, otherwise it proceeds with another pass through the network.

In this architecture a single SVM never has to deal with the whole training set. If the filters in the first few layers are efficient in extracting the support vectors then the largest optimization, the one of the last layer, has to handle only a few more vectors than the number of actual support vectors. Therefore, in problems where the support vectors are a small subset of the training vectors - which is usually the case - each of the sub-problems is much smaller than the whole problem (compare section 4).

## 2.1 Notation (2-class, maximum margin)

We discuss here the 2-class classification problem, solved in dual formulation. The Cascade does not depend on details of the optimization algorithm and alternative formulations or regression algorithms map equally well onto this architecture. The 2-class problem is the most difficult one to parallelize because there is no natural split into sub-problems. Multi-class problems can always be separated into 2-class problems.

Let us consider a set of $l$ training examples $(x_i;\ y_i)$; where $x_i \in R^d$ represents a d-dimensional pattern and $y_i = \pm 1$ the class label. $K(x_i,x_j)$ is the matrix of kernel values between patterns and $\alpha_i$ the Lagrange coefficients to be determined by the optimization. The SVM solution for this problem consists in maximizing the following quadratic optimization function (dual formulation):

$$\max\ W(\alpha) = -1/2 * \sum_i^l \sum_j^l \alpha_i \alpha_j y_i y_j K(x_i, x_j) + \sum_i^l \alpha_i \qquad (1)$$

$$\text{Subject to:}\ \ 0 \le \alpha_i \le C,\ \forall i \quad \text{and} \quad \sum_i^l \alpha_i y_i = 0$$

The gradient $\mathbf{G} = \nabla W(\alpha)$ of W with respect to $\boldsymbol{\alpha}$ is then:

$$G_i = \frac{\partial W}{\partial \alpha_i} = -y_i \sum_{j=1}^l y_j \alpha_j K(x_i, x_j) + 1 \qquad (2)$$

## 2.2 Formal proof of convergence

The main issue is whether a Cascade architecture will actually converge to the global optimum. The following theorems show that this is the case for a wide range of conditions. Let $S$ denote a subset of the training set $\Omega$, W($S$) is the optimal objective function over $S$ (equation 1), and let $Sv(S) \subset S$ be the subset of S for which the optimal $\alpha$ are non-zero (support vectors of S). It is obvious that:

$$\forall S \subset \Omega, \ W(S) = W(Sv(S)) \leq W(\Omega)$$

$$(3)$$

Let us consider a family F of sets of training examples for which we can independently compute the SVM solution. The set $S^* \in F$ that achieves the greatest W(S) will be called the best set in family F. We will write W(F) as a shorthand for W(S*), that is:

$$W(F) = \max_{S \in F} W(S) \leq W(\Omega) \qquad (4)$$

We are interested in defining a sequence of families $F_t$ such that $W(F_t)$ converges to the optimum. Two results are relevant for proving convergence.

**Theorem 1:** Let us consider two families F and G of subsets of $\Omega$. If a set $T \in G$ contains the support vectors of the best set $S_F^* \in F$, then $W(G) \geq W(F)$.

Proof: Since $Sv(S_F^*) \subset T$, we have $W(S_F^*) = W(Sv(S_F^*)) \leq W(T)$. Therefore, $W(F) = W(S_F^*) \leq W(T) \leq W(G)$

**Theorem 2:** Let us consider two families F and G of subsets of $\Omega$. Assume that every set $T \in G$ contains the support vectors of the best set $S_F^* \in F$.

$$If \ \ W(G) = W(F) \Rightarrow W(S_F^*) = W(\bigcup_{T \in G} T).$$

Proof: Theorem 1 implies that $W(G) \geq W(F)$. Consider a vector $\alpha^*$ solution of the SVM problem restricted to the support vectors $Sv(S_F^*)$. For all $T \in G$, we have $W(T) \geq W(Sv(S_F^*))$ because $Sv(S_F^*)$ is a subset of T. We also have $W(T) \leq W(G) = W(F) = W(S_F^*) = W(Sv(S_F^*))$. Therefore $W(T) = W(Sv(S_F^*))$. This implies that $\alpha^*$ is also a solution of the SVM on set T. Therefore $\alpha^*$ satisfies all the KKT conditions corresponding to all sets $T \in G$. This implies that $\alpha^*$ also satisfies the KKT conditions for the union of all sets in G.

**Definition 1.** A Cascade is a sequence $(F_t)$ of families of subsets of $\Omega$ satisfying:
 i) For all t > 1, a set $T \in F_t$ contains the support vectors of the best set in $F_{t-1}$.
 ii) For all t, there is a k > t such that:
- All sets $T \in F_k$ contain the support vectors of the best set in $F_{k-1}$.
- The union of all sets in $F_k$ is equal to $\Omega$.

**Theorem 3:** A Cascade $(F_t)$ converges to the SVM solution of $\Omega$ in finite time, namely: $\exists t^*, \forall t > t^*, W(F_t) = W(\Omega)$

Proof: Assumption i) of Definition 1 plus theorem 1 imply that the sequence $W(F_t)$ is monotonically increasing. Since this sequence is bounded by $W(\Omega)$, it converges to some value $W^* \leq W(\Omega)$. The sequence $W(F_t)$ takes its values in the finite set of the W(S) for all $S \subset \Omega$. Therefore there is a $l > 0$ such that $\forall t > l, \ W(F_t) = W^*$. This observation, assertion ii) of definition 1, plus theorem 2 imply that there is a k > $l$ such that $W(F_k) = W(\Omega)$. Since $W(F_t)$ is monotonically increasing, $W(F_k) = W(\Omega)$ for all t > k.

As stated in theorem 3, a layered Cascade architecture is guaranteed to converge to the global optimum if we keep the best set of support vectors produced in one layer, and use it in at least one of the subsets in the next layer. This is the case in the binary Cascade shown in Figure 1. However, not all layers meet assertion *ii)* of Definition 1. The union of sets in a layer is not equal to the whole training set, except in the first layer. By introducing the feedback loop that enters the result of the last layer into the

first one, combined with all non-support vectors, we fulfill all assertions of Definition 1. We can test for global convergence in layer 1 and do a fast filtering in the subsequent layers.

## 2.3  Interpretation of the SVM filtering process

An intuitive picture of the filtering process is provided in Figure 2. If a subset $S \subset \Omega$ is chosen randomly from the training set, it will most likely not contain all support vectors of $\Omega$ and its support vectors may not be support vectors of the whole problem. However, if there is not a serious bias in a subset, support vectors of S are likely to contain some support vectors of the whole problem. Stated differently, it is plausible that 'interior' points in a subset are going to be 'interior' points in the whole set. Therefore, a non-support vector of a subset has a good chance of being a non-support vector of the whole set and we can eliminate it from further analysis.

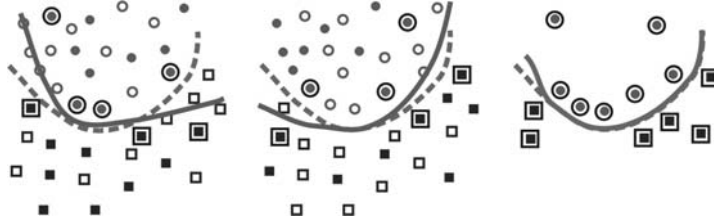

**Figure 2:** A toy problem illustrating the filtering process. Two disjoint subsets are selected from the training data and each of them is optimized individually (left, center; the data selected for the optimizations are the solid elements). The support vectors in each of the subsets are marked with frames. They are combined for the final optimization (right), resulting in a classification boundary (solid curve) close to the one for the whole problem (dashed curve).

# 3  Distributed Optimization

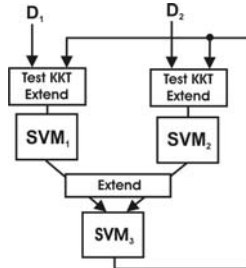

$$W_i = -\frac{1}{2}\vec{\alpha}_i^T \, Q_i \vec{\alpha}_i + \vec{e}_i^{\,T} \vec{\alpha}_i;$$

$$\vec{G}_i = -\vec{\alpha}_i^T \, Q_i + \vec{e}_i;$$

(5)

**Figure 3:** A Cascade with two input sets $D_1$, $D_2$. $W_i$, $G_i$ and $Q_i$ are objective function, gradient, and kernel matrix, respectively, of $SVM_i$ (in vector notation); $e_i$ is a vector with all 1. Gradients of $SVM_1$ and $SVM_2$ are merged (Extend) as indicated in (6) and are entered into $SVM_3$. Support vectors of $SVM_3$ are used to test $D_1$, $D_2$ for violations of the KKT conditions. Violators are combined with the support vectors for the next iteration.

Section 2 shows that a distributed architecture like the Cascade indeed converges to the global solution, but no indication is provided how efficient this approach is. For a good performance we try to advance the optimization as much as possible in each stage. This depends on how the data are split initially, how partial results are merged and how well an optimization can start from the partial results provided by the previous stage. We focus on gradient-ascent algorithms here, and discuss how to handle merging efficiently.

### 3.1 Merging subsets

For this discussion we look at a Cascade with two layers (Figure 3). When merging the two results of $SVM_1$ and $SVM_2$, we can initialize the optimization of $SVM_3$ to different starting points. In the general case the merged set starts with the following optimization function and gradient:

$$W_3 = -\frac{1}{2}\begin{bmatrix} \vec{\alpha}_1 \\ \vec{\alpha}_2 \end{bmatrix}^T \begin{bmatrix} Q_1 & Q_{12} \\ Q_{21} & Q_2 \end{bmatrix}\begin{bmatrix} \vec{\alpha}_1 \\ \vec{\alpha}_2 \end{bmatrix} + \begin{bmatrix} \vec{e}_1 \\ \vec{e}_2 \end{bmatrix}^T\begin{bmatrix} \vec{\alpha}_1 \\ \vec{\alpha}_2 \end{bmatrix} \quad \vec{G}_3 = -\begin{bmatrix} \vec{\alpha}_1 \\ \vec{\alpha}_2 \end{bmatrix}^T \begin{bmatrix} Q_1 & Q_{12} \\ Q_{21} & Q_2 \end{bmatrix} + \begin{bmatrix} \vec{e}_1 \\ \vec{e}_2 \end{bmatrix} \quad (6)$$

We consider two possible initializations:

Case 1: $\vec{\alpha}_1 = \vec{\alpha}_1 \ of \ SVM_1; \vec{\alpha}_2 = \vec{0}$;

Case 2: $\vec{\alpha}_1 = \vec{\alpha}_1 \ of \ SVM_1; \ \vec{\alpha}_2 = \vec{\alpha}_2 \ of \ SVM_2$. $\quad (7)$

Since each of the subsets fulfills the KKT conditions, each of these cases represents a feasible starting point with: $\sum \alpha_i y_i = 0$.

Intuitively one would probably assume that case 2 is the preferred one since we start from a point that is optimal in the two spaces defined by the vectors $D_1$ and $D_2$. If $Q_{12}$ is 0 ($Q_{21}$ is then also 0 since the kernel matrix is symmetric), the two spaces are orthogonal (in feature space) and the sum of the two solutions is the solution of the whole problem. Therefore, case 2 is indeed the best choice for initialization, because it represents the final solution. If, on the other hand, the two subsets are identical, then an initialization with case 1 is optimal, since this represents now the solution of the whole problem. In general, we are probably somewhere between these two cases and therefore it is not obvious, which case is best.

While the theorems of section 2 guarantee the convergence to the global optimum, they do not provide any indication how fast this going to happen. Empirically we find that the Cascade converges quickly to the global solution, as is indicated in the examples below. All the problems we tested converge in 2 to 5 passes.

## 4 Experimental results

We implemented the Cascade architecture for a single processor as well as for a cluster of processors and tested it extensively with several problems; the largest are: MNIST[1], FOREST[2], NORB[3] (all are converted to 2-class problems). One of the main advantages of the Cascade architecture is that it requires far less memory than a single SVM, because the size of the kernel matrix scales with the square of the active set. This effect is illustrated in Figure 4. It has to be emphasized that both cases, single SVM and Cascade, use shrinking, but shrinking alone does not solve the problem of exorbitant sizes of the kernel matrix.

A good indication of the Cascade's inherent efficiency is obtained by counting the number of kernel evaluations required for one pass. As shown in Table 1, a 9-layer Cascade requires only about 30% as many kernel evaluations as a single SVM for

100,000 training vectors. How many kernel evaluations actually have to be computed depends on the caching strategy and the memory size.

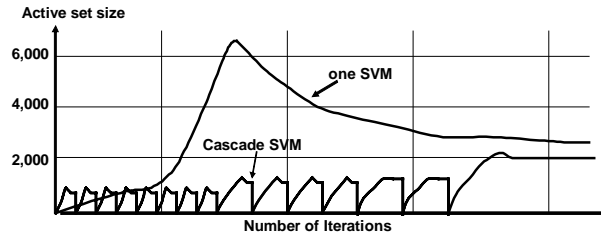

**Figure 4:** The size of the active set as a function of the number of iterations for a problem with 30,000 training vectors. The upper curve represents a single SVM, while the lower one shows the active set size for a 4-layer Cascade.

As indicated in Table 1, this parameter, and with it the compute times, are reduced even more. Therefore, even a simulation on a single processor can produce a speed-up of 5x to 10x or more, depending on the available memory size. For practical purposes often a single pass through the Cascade produces sufficient accuracy (compare Figure 5). This offers a particularly simple way for solving problems of a size that would otherwise be out of reach for SVMs.

| Number of Layers | 1 | 2 | 3 | 4 | 5 | 6 | 7 | 8 | 9 |
|---|---|---|---|---|---|---|---|---|---|
| K-eval request x$10^9$ | 106 | 89 | 77 | 68 | 61 | 55 | 48 | 42 | 38 |
| K-eval x$10^9$ | 33 | 12 | 4.5 | 3.9 | 2.7 | 2.4 | 1.9 | 1.6 | 1.4 |

**Table 1:** Number of Kernel evaluations (requests and actual, with a cache size of 800MB) for different numbers of layers in the Cascade (single pass). The number of Kernel evaluations is reduced as the number of Cascade layers increases. Then, larger amounts of the problems fit in the cache, reducing the actual Kernel computations even more. Problem: FOREST, 100K vectors.

| Iteration | Training time | Max # training vect. per machine | # Support Vectors | W | Acc. |
|---|---|---|---|---|---|
| 0 | 21.6h | 72,658 | 54,647 | 167427 | 99.08% |
| 1 | 22.2h | 67,876 | 61,084 | 174560 | 99.14% |
| 2 | 0.8h | 61,217 | 61,102 | 174564 | 99.13% |

**Table 2:** Training times for a large data set with 1,016,736 vectors (MNIST was expanded by warping the handwritten digits). A Cascade with 5 layers is executed on a Linux cluster with 16 machines (AMD 1800, dual processors, 2GB RAM per machine). The solution converges in 3 iterations. Shown are also the maximum number of training vectors on one machine and the number of support vectors in the last stage. W: optimization function; Acc: accuracy on test set. Kernel: RBF, gamma=1; C=50.

Table 2 shows how a problem with over one million vectors is solved in about a day (single pass) with a generalization performance equivalent to the fully converged solution. While the full training set contains over 1M vectors, one processor never handles more than 73k vectors in the optimization and 130k for the convergence test. The Cascade provides several advantages over a single SVM because it can reduce compute- as well as storage-requirements. The main limitation is that the last layer consists of one single optimization and its size has a lower limit given by the number of support vectors. This is why the acceleration saturates at a relatively small number

of layers. Yet this is not a hard limit since a single optimization can be distributed over multiple processors as well, and we are working on efficient implementations of such algorithms.

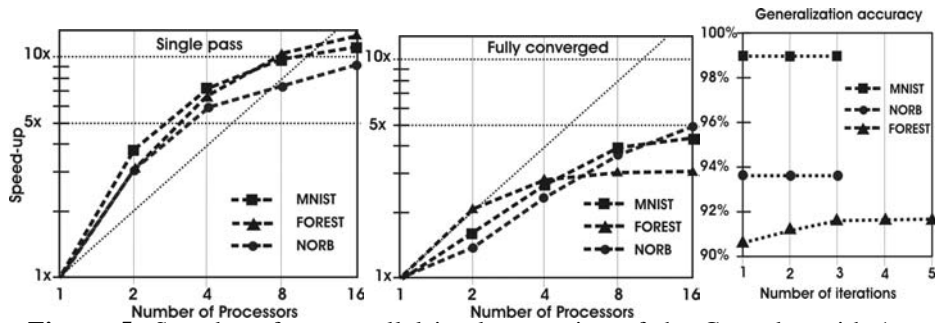

**Figure 5:** Speed-up for a parallel implementation of the Cascades with 1 to 5 layers (1 to 16 SVMs, each running on a separate processor), relative to a single SVM: single pass (left), fully converged (middle) (MNIST, NORB: 3 iterations, FOREST: 5 iterations). On the right is the generalization performance of a 5-layer Cascade, measured after each iteration. For MNIST and NORB, the accuracy after one pass is the same as after full convergence (3 iterations). For FOREST, the accuracy improves from 90.6% after a single pass to 91.6% after convergence (5 iterations). Training set sizes: MNIST: 60k, NORB: 48k, FOREST: 186k.

## Footnotes

[1] MNIST: handwritten digits, d=784 (28x28 pixels); training: 60,000; testing: 10,000; classes: odd digits - even digits;  http://yann.lecun.com/exdb/mnist.

[2] FOREST: d=54; class 2 versus rest; training: 560,000; testing: 58,100 ftp://ftp.ics.uci.edu/pub/machine-learning-databases/covtype/covtype.info.

[3] NORB: images, d=9,216 ; training$_r$=48,600; testing=48,600; monocular; merged class 0 and 1 versus the rest. http://www.cs.nyu.edu/~ylclab/data/norb-v1.0

## References

[1] V. Vapnik, "Statistical Learning Theory", Wiley, New York, 1998.

[2] B. Boser, I. Guyon, V. Vapnik, "A training algorithm for optimal margin classifiers" in Proc. 5th Annual Workshop on Computational Learning Theory, Pittsburgh, ACM, 1992.

[3] E. Osuna, R. Freund, F. Girosi, "Training Support Vector Machines, an Application to Face Detection", in Computer vision and Pattern Recognition, pp.130-136, 1997.

[4] T. Joachims, "Making large-scale support vector machine learning practical", in Advances in Kernel Methods, B. Schölkopf, C. Burges, A. Smola, (eds.), Cambridge, MIT Press, 1998.

[5] J.C. Platt, "Fast training of support vector machines using sequential minimal optimization", in Adv. in Kernel Methods: Schölkopf, C. Burges, A. Smola (eds.), 1998.

[6] C. Chang, C. Lin, "LIBSVM", http://www.csie.ntu.edu.tw/~cjlin/libsvm/.

[7] R. Collobert, S. Bengio, and J. Mariéthoz. Torch: A modular machine learning software library. Technical Report IDIAP-RR 02-46, IDIAP, 2002.

[8] D. DeCoste and B. Schölkopf, "Training Invariant Support Vector Machines", Machine Learning, 46, 161-190, 2002.

[9] R. Collobert, Y. Bengio, S. Bengio, "A Parallel Mixture of SVMs for Very Large Scale Problems", in Neural Information Processing Systems, Vol. 17, MIT Press, 2004.

[10] A. Tveit, H. Engum. Parallelization of the Incremental Proximal Support Vector Machine Classifier using a Heap-based Tree Topology. Tech. Report, IDI, NTNU, Trondheim, 2003.

[11] J. X. Dong, A. Krzyzak , C. Y. Suen, "A fast Parallel Optimization for Training Support Vector Machine." *Proceedings of 3rd International Conference o*n *Machine Learning and Data Mining*, P. Perner and A. Rosenfeld (Eds.) Springer Lecture Notes in Artificial Intelligence (LNAI 2734), pp. 96--105, Leipzig, Germany, July 5-7, 2003.

[12] G. Zanghirati, L. Zanni, "A parallel solver for large quadratic programs in training support vector machines", Parallel Computing, Vol. 29, pp.535-551, 2003.

